# Random Sampling of States in Dynamic Programming

**Christopher G. Atkeson and Benjamin Stephens**
Robotics Institute, Carnegie Mellon University
cga@cmu.edu, bstephens@cmu.edu
www.cs.cmu.edu/∼cga, www.cs.cmu.edu/∼bstephe1

## Abstract

We combine three threads of research on approximate dynamic programming: sparse random sampling of states, value function and policy approximation using local models, and using local trajectory optimizers to globally optimize a policy and associated value function. Our focus is on finding steady state policies for deterministic time invariant discrete time control problems with continuous states and actions often found in robotics. In this paper we show that we can now solve problems we couldn't solve previously.

## 1  Introduction

Optimal control provides a potentially useful methodology to design nonlinear control laws (policies) $\mathbf{u} = \mathbf{u}(\mathbf{x})$ which give the appropriate action $\mathbf{u}$ for any state $\mathbf{x}$. Dynamic programming provides a way to find globally optimal control laws, given a one step cost (a.k.a. "reward" or "loss") function and the dynamics of the problem to be optimized. We focus on control problems with continuous states and actions, deterministic time invariant discrete time dynamics $\mathbf{x}_{k+1} = \mathbf{f}(\mathbf{x}_k, \mathbf{u}_k)$, and a time invariant one step cost function $L(\mathbf{x}, \mathbf{u})$. Policies for such time invariant problems will also be time invariant. We assume we know the dynamics and one step cost function. Future work will address simultaneously learning a dynamic model, finding a robust policy, and performing state estimation with an erroneous partially learned model. One approach to dynamic programming is to approximate the value function $V(\mathbf{x})$ (the optimal total future cost from each state $V(\mathbf{x}) = \sum_{k=0}^{\infty} L(\mathbf{x}_k, \mathbf{u}_k)$), and to repeatedly solve the Bellman equation $V(\mathbf{x}) = \min_{\mathbf{u}}(L(\mathbf{x}, \mathbf{u}) + V(\mathbf{f}(\mathbf{x}, \mathbf{u})))$ at sampled states $\mathbf{x}$ until the value function estimates have converged to globally optimal values. We explore approximating the value function and policy using many local models.

**An example problem:** We use one link pendulum swingup as an example problem in this introduction to provide the reader with a visualizable example of a value function and policy. In one link pendulum swingup a motor at the base of the pendulum swings a rigid arm from the downward stable equilibrium to the upright unstable equilibrium and balances the arm there (Figure 1). What makes this challenging is that the one step cost function penalizes the amount of torque used and the deviation of the current position from the goal. The controller must try to minimize the total cost of the trajectory. The one step cost function for this example is a weighted sum of the squared position errors ($\theta$: difference between current angles and the goal angles) and the squared torques $\tau$: $L(\mathbf{x}, \mathbf{u}) = 0.1\theta^2 \mathrm{T} + \tau^2 \mathrm{T}$ where 0.1 weights the position error relative to the torque penalty, and T is the time step of the simulation (0.01s). There are no costs associated with the joint velocity. Figure 2 shows the value function and policy generated by dynamic programming.

One important thread of research on approximate dynamic programming is developing representations that adapt to the problem being solved and extend the range of problems that can be solved with a reasonable amount of memory and time. Random sampling of states has been proposed by a number of researchers [1, 2, 3, 4, 5, 6, 7]. In our case we add new randomly selected states as we

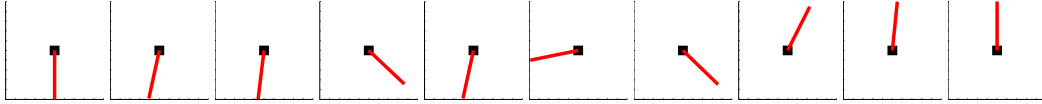

Figure 1: Configurations from the simulated one link pendulum optimal trajectory every half a second and at the end of the trajectory.

solve the problem, allowing the "grid" that results to reflect the local complexity of the value function as we generate it. Figure 2:right shows such a randomly generated set of states superimposed on a contour plot of the value function for one link swingup.

Another important thread in our work on applied dynamic programming is developing ways for grids or random samples to be as sparse as possible. One technique that we apply here is to represent full trajectories from each sampled state to the goal, and to refine each trajectory using local trajectory optimization [8]. Figure 2:right shows a set of optimized trajectories from the sampled states to the goal. One key aspect of the local trajectory optimizer we use is that it provides a local quadratic model of the value function and a local linear model of the policy at the sampled state. These local models help our function approximators handle sparsely sampled states. To obtain globally optimal solutions, we incorporate exchange of information between non-neighboring sampled states.

**On what problems will the proposed approach work?** We believe our approach can discover underlying simplicity in many typical problems. An example of a problem that appears complex but is actually simple is a problem with linear dynamics and a quadratic one step cost function. Dynamic programming can be done for linear quadratic regulator (LQR) problems even with hundreds of dimensions and it is not necessary to build a grid of states [9]. The cost of representing the value function is quadratic in the dimensionality of the state. The cost of performing a "sweep" or update of the value function is at most cubic in the state dimensionality. Continuous states and actions are easy to handle. Perhaps many problems, such as the examples in this paper, have simplifying characteristics similar to LQR problems. For example, problems that are only "slightly" nonlinear and have a locally quadratic cost function may be solvable with quite sparse representations. One goal of our work is to develop methods that do not immediately build a hugely expensive representation if it is not necessary, and attempt to harness simple and inexpensive parallel local planning to solve complex planning problems. Another goal of our work is to develop methods that can take advantage of situations where only a small amount of global interaction is necessary to enable local planners capable of solving local problems to find globally optimal solutions.

## 2   Related Work

**Random state selection:** Random grids and random sampling are well known in numerical integration, finite element methods, and partial differential equations. Rust applied random sampling of states to dynamic programming [1, 10]. He showed that random sampling of states can avoid the curse of dimensionality for stochastic dynamic programming problems with a finite set of discrete actions. This theoretical result focused on the cost of computing the expectation term in the stochastic version of the Bellman equation. [11] claim the assumptions used in [1] are unrealistically restrictive, and [12] point out that the complexity of Rust's approach is proportional to the Lipschitz constant of the problem data, which often increases exponentially with increasing dimensions. The practicality and usefulness of random sampling of states in deterministic dynamic programming with continuous actions (the focus of our paper) remains an open question. We note that deterministic problems are usually more difficult to solve since the random element in the stochastic dynamics smooths the dynamics and makes them easier to sample. Alternatives to random sampling of states are irregular or adaptive grids [13], but in our experience they still require too many representational resources as the problem dimensionality increases.

In reinforcement learning random sampling of states is sometimes used to provide training data for function approximation of the value function. Reinforcement learning also uses random exploration for several purposes. In model-free approaches exploration is used to find actions and states that lead to better outcomes. This process is somewhat analogous to the random state sampling described in this paper for model-based approaches. In model-based approaches, exploration is also used to improve the model of the task. In our paper it is assumed a model of the task is available, so this type of exploration is not necessary.

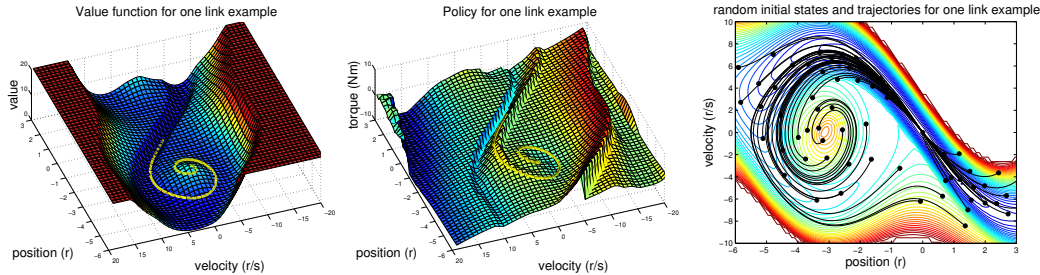

Figure 2: **Left and Middle:** The value function and policy for a one link pendulum swingup. The optimal trajectory is shown as a yellow line in the value function plot, and as a black line with a yellow border in the policy plot. The value function is cut off above 20 so we can see the details of the part of the value function that determines the optimal trajectory. The goal is at the state (0,0). **Right:** Random states (dots) and trajectories (black lines) used to plan one link swingup, superimposed on a contour map of the value function.

In the field of Partially Observable Markov Decision Processes (POMDPs) there has been some work on randomly sampling belief states, and also using local models of the value function and its first derivative at each randomly sampled belief state (for example [2, 3, 4, 5, 6, 7]). Thrun explored random sampling of belief states where the underlying states and actions were continuous [7]. He used a nearest neighbor scheme to perform value function interpolation, and a coverage test to decide whether to accept a new random state (is a new random state far enough from existing states?) rather than a surprise test (is the value of the new random state predicted incorrectly?).

In robot planning for obstacle avoidance random sampling of states is now quite popular [14]. Probabilistic Road Map (PRM) methods build a graph of plans between randomly selected states. Rapidly Exploring Random Trees (RRTs) grow paths or trajectories towards randomly selected states. In general it is difficult to modify PRM and RRT approaches to find optimal paths, and the resulting algorithms based on RRTs are very similar to A* search.

## 3   Combining Random State Sampling With Local Optimization

The process of using the Bellman equation to update a representation of the value function by minimizing over all actions at a state is referred to as value iteration. Standard value iteration represents the value function and associated policy using multidimensional tables, with each entry in the table corresponding to a particular state. In our approach we randomly select states, and associate with each state a local quadratic model of the value function and a local linear model of the policy. Our approach generalizes value iteration, and has the following components: 1. There is a "global" function approximator for both the value function and the policy. In our current implementation the value function and policy are represented through a combination of sampled and parametric representations, building global approximations by combining local models. 2. It is possible to estimate the value of a state in two ways. The first is to use the approximated value function. The second is our analog of using the Bellman equation: use the cost of a trajectory starting from the state under consideration and following the current global policy. The trajectory is optimized using local trajectory optimization. 3. As in a Bellman update, there is a way to globally optimize the value of a state by considering many possible "actions". In our approach we consider many local policies associated with different stored states.

**Taking advantage of goal states:** For problems with goal states there are several ways to speed up convergence. In cases where LQR techniques apply [9], we use the policy obtained by solving the corresponding LQR control problem at the goal as the default policy everywhere, to which the policy computed by dynamic programming is added. [15] plots an example of a default policy and the policy generated by dynamic programming for comparison. We limit the outputs of this default policy. In setting up the goal LQR controller, a radius is established and tested within which the goal LQR controller always works and achieves close to the predicted optimal cost. This has the effect of making of enlarging the goal. If the dynamic programming process can get within the LQR radius of the goal, it can use only the default policy to go the rest of the way. If it is not possible to create a goal LQR controller due to a hard nonlinearity, or if there is no goal state, it does not have to be done as the goal controller merely accelerates the solution process. The proposed technique can be generalized in a straightforward way to use any default goal policy. In this paper the swingup

problems use an LQR default policy, which was limited for each action dimension to ±5Nm. For the balance problem we did not use a default policy. We note that for the swingup problems shown here the default LQR policy is capable of balancing the inverted pendulum at the goal, but is not capable of swinging up the pendulum to the goal.

We also initially only generate the value function and policy in the region near the goal. This solved region is gradually increased in size by increasing a value function threshold. Examples of regions bounded by a constant value are shown by the value function contours in Figure 2. [16] describes how to handle periodic tasks which have no goal states, and also discontinuities in the dynamics.

**Local models of the value function and policy:** We need to represent value functions as sparsely as possible. We propose a hybrid tabular and parametric approach: parametric local models of the value function and policy are represented at sampled locations. This representation is similar to using many Taylor series approximations of a function at different points. At each sampled state $\mathbf{x}^p$ the local quadratic model for the value function is:

$$V^p(\mathbf{x}) \approx V_0^p + V_x^p \hat{\mathbf{x}} + \frac{1}{2}\hat{\mathbf{x}}^{\mathrm{T}} V_{xx}^p \hat{\mathbf{x}} \tag{1}$$

where $\hat{\mathbf{x}} = \mathbf{x} - \mathbf{x}^p$ is the vector from the stored state $\mathbf{x}^p$, $V_0^p$ is the constant term of the local model, $V_x^p$ is the first derivative of the local model (and the value function) at $\mathbf{x}^p$, and $V_{xx}^p$ is the second derivative of the local model (and the value function) at $\mathbf{x}^p$. The local linear model for the policy is:

$$\mathbf{u}^p(\mathbf{x}) = \mathbf{u}_0^p - \mathbf{K}^p \hat{\mathbf{x}} \tag{2}$$

where $\mathbf{u}_0^p$ is the constant term of the local policy, and $\mathbf{K}^p$ is the first derivative of the local policy and also the gain matrix for a local linear controller.

**Creating the local model:** These local models of the value function can be created using Differential Dynamic Programming (DDP) [17, 18, 8, 16]. This local trajectory optimization process is similar to linear quadratic regulator design in that a local model of the value function is produced. In DDP, value function and policy models are produced at each point along a trajectory. Suppose at a point $(\mathbf{x}^i, \mathbf{u}^i)$ we have 1) a local second order Taylor series approximation of the optimal value function: $V^i(\mathbf{x}) \approx V_0^i + V_x^i \hat{\mathbf{x}} + \frac{1}{2}\hat{\mathbf{x}}^{\mathrm{T}} V_{xx}^i \hat{\mathbf{x}}$ where $\hat{\mathbf{x}} = \mathbf{x} - \mathbf{x}^i$. 2) a local second order Taylor series approximation of the robot dynamics, which can be learned using local models of the dynamics ($\mathbf{f}_x^i$ and $\mathbf{f}_u^i$ correspond to $\mathbf{A}$ and $\mathbf{B}$ of the linear plant model used in linear quadratic regulator (LQR) design): $\mathbf{x}_{k+1} = \mathbf{f}^i(\mathbf{x}, \mathbf{u}) \approx \mathbf{f}_0^i + \mathbf{f}_x^i \hat{\mathbf{x}} + \mathbf{f}_u^i \hat{\mathbf{u}} + \frac{1}{2}\hat{\mathbf{x}}^{\mathrm{T}} \mathbf{f}_{xx}^i \hat{\mathbf{x}} + \hat{\mathbf{x}}^{\mathrm{T}} \mathbf{f}_{xu}^i \hat{\mathbf{u}} + \frac{1}{2}\hat{\mathbf{u}}^{\mathrm{T}} \mathbf{f}_{uu}^i \hat{\mathbf{u}}$ where $\hat{\mathbf{u}} = \mathbf{u} - \mathbf{u}^i$, and 3) a local second order Taylor series approximation of the one step cost, which is often known analytically for human specified criteria ($L_{xx}$ and $L_{uu}$ correspond to $\mathbf{Q}$ and $\mathbf{R}$ of LQR design): $L^i(\mathbf{x}, \mathbf{u}) \approx L_0^i + L_x^i \hat{\mathbf{x}} + L_u^i \hat{\mathbf{u}} + \frac{1}{2}\hat{\mathbf{x}}^{\mathrm{T}} L_{xx}^i \hat{\mathbf{x}} + \hat{\mathbf{x}}^{\mathrm{T}} L_{xu}^i \hat{\mathbf{u}} + \frac{1}{2}\hat{\mathbf{u}}^{\mathrm{T}} L_{uu}^i \hat{\mathbf{u}}$

Given a trajectory, one can integrate the value function and its first and second spatial derivatives backwards in time to compute an improved value function and policy. We utilize the "Q function" notation from reinforcement learning: $Q(\mathbf{x}, \mathbf{u}) = L(\mathbf{x}, \mathbf{u}) + V(\mathbf{f}(\mathbf{x}, \mathbf{u}))$. The backward sweep takes the following form (in discrete time):

$$Q_x^i = L_x^i + V_x^i \mathbf{f}_x^i; \quad Q_u^i = L_u^i + V_x^i \mathbf{f}_u^i \tag{3}$$

$$Q_{xx}^i = L_{xx}^i + V_x^i \mathbf{f}_{xx}^i + (\mathbf{f}_x^i)^{\mathrm{T}} V_{xx}^i \mathbf{f}_x^i; \quad Q_{ux}^i = L_{ux}^i + V_x^i \mathbf{f}_{ux}^i + (\mathbf{f}_u^i)^{\mathrm{T}} V_{xx}^i \mathbf{f}_x^i; \quad Q_{uu}^i = L_{uu}^i + V_x^i \mathbf{f}_{uu}^i + (\mathbf{f}_u^i)^{\mathrm{T}} V_{xx}^i \mathbf{f}_u^i \tag{4}$$

$$\Delta \mathbf{u}^i = (Q_{uu}^i)^{-1} Q_u^i; \quad \mathbf{K}^i = (Q_{uu}^i)^{-1} Q_{ux}^i \tag{5}$$

$$V_x^{i-1} = Q_x^i - Q_u^i \mathbf{K}^i; \quad V_{xx}^{i-1} = Q_{xx}^i - Q_{xu}^i \mathbf{K}^i \tag{6}$$

where subscripts indicate derivatives and superscripts indicate the trajectory index. After the backward sweep, forward integration can be used to update the trajectory itself: $\mathbf{u}_{new}^i = \mathbf{u}^i - \Delta \mathbf{u}^i - \mathbf{K}^i(\mathbf{x}_{new}^i - \mathbf{x}^i)$. We note that the cost of this approach grows at most cubically rather than exponentially with respect to the dimensionality of the state.

In problems that have a goal state, we can generate a trajectory from each stored state all the way to the goal. The cost of this trajectory is an upper bound on the true value of the state, and is used to bound the estimated value for the old state.

**Utilizing the local models:** For the purpose of explaining our algorithm, let's assume we already have a set of sampled states, each of which has a local model of the value function and the policy.

How should we use these multiple local models? The simplest approach is to just use the predictions of the nearest sampled state, which is what we currently do. We use a kd-tree to efficiently find nearest neighbors, but there are many other approaches that will find nearby stored states efficiently. In the future we will investigate using other methods to combine local model predictions from nearby stored states: distance weighted averaging (kernel regression), linear locally weighted regression, and quadratic locally weighted regression for value functions.

**Creating new random states:** For tasks with a goal state, we initialize the set of stored states by storing the goal state itself. We have explored a number of distributions to select additional states from: uniform within bounds on the states; Gaussian with the mean at the goal; sampling near existing states; and sampling from an underlying low resolution regular grid. The uniform approach is a useful default approach, which we use in the swingup examples, the Gaussian approach provides a simple way to tune the distribution, sampling near existing states provides a way to efficiently sample while growing the solved region in high dimensions, and sampling from an underlying low resolution grid seems to perform well when only a small number of stored states are used (similar to using low dispersion sequences [1, 14]). A key point of our approach is that we do not generate the random states in advance but instead select them as the algorithm progresses. This allows us to apply an acceptance criteria to candidate states, which we describe in the next paragraph. We have also explored changing the distribution we generate candidate states from as the algorithm progresses, for example using a mixture of Gaussians with the Gaussians centered on existing stored states. Another reasonable hybrid approach would be to initially sample from a grid, and then bias more general sampling to regions of higher value function approximation error.

**Acceptance criteria for candidate states:** We have several criteria to accept or reject states to be permanently stored. In the future we will explore "forgetting" or removing stored states, but at this point we apply all memory control techniques at the storage event. To focus the search and limit the volume considered, a steadily increasing value limit is maintained ($V_{limit}$), which is increased slightly after each use. The approximated value function is used to predict the value of the candidate state. If the prediction is above $V_{limit}$, the candidate state is rejected. Otherwise, a trajectory is created from the candidate state using the current approximated policy, and then locally optimized. If the value of that trajectory is above $V_{limit}$, the candidate state is rejected. If the value of the trajectory is within 10% of the predicted value, the candidate state is rejected. Only "surprises" are stored. For problems with a goal state, if the trajectory does not reach the goal the candidate state is rejected. Other criteria such as an A* like criteria (cost-to-go($\mathbf{x}$) + cost-from-start($\mathbf{x}$) > *threshold*) can be used to reject candidate states. All of the thresholds mentioned can be changed as the algorithm progresses. For example, $V_{limit}$ is gradually increased during the solution process, to increase the volume considered by the algorithm. We currently use a 10% "surprise" threshold. In future work we will explore starting with a larger threshold and decreasing this threshold with time, to further reduce the number of samples accepted and stored while improving convergence. It is possible to take the distance to the nearest sampled state into account in the acceptance criteria for new samples. The common approach of accepting states beyond a distance threshold enforces a minimum resolution, and leads to potentially severe curse of dimensionality effects. Rejecting states that are too close to existing states will increase the error in representing the value function, but may be a way for preventing too many samples near complex regions of the value functions that have little practical effect. For example, we often do not need much accuracy in representing the value function near policy discontinuities where the value function has discontinuities in its spatial derivative and "creases". In these areas the trajectories typically move away from the discontinuities, and the details of the value function have little effect.

In the current implementation, after a candidate state is accepted, the state in the database whose local model was used to make the prediction is re-optimized including information from the newly added point, since the prediction was wrong and the new point's policy may lead to a better value for that state.

**Creating a trajectory from a state:** We create a trajectory from a candidate state or refine a trajectory from a stored state in the same way. The first step is to use the current approximated policy until the goal or a time limit is reached. In the current implementation this involves finding the stored state nearest to the current state in the trajectory and using its locally linear policy to compute the action on each time step. The second step is to locally optimize the trajectory. We use Differential Dynamic Programming (DDP) in the current implementation [17, 18, 8, 16]. In the current implementation we do not save the trajectory but only the local models from its start. If the cost of the

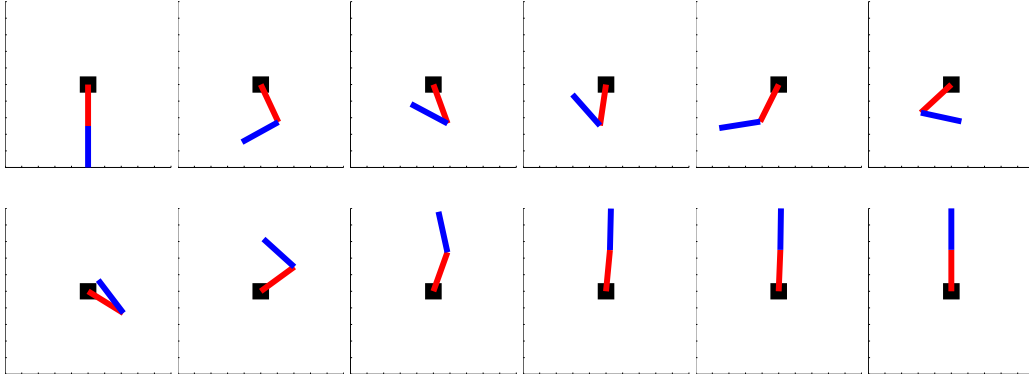

Figure 3: Configurations from the simulated two link pendulum optimal swing up trajectory every fifth of a second and the end of the trajectory.

trajectory is more than the currently stored value for the state, we reject the new value, as the values all come from actual trajectories and are upper bounds for the true value. We always keep the lowest upper bound.

**Combining parallel greedy local optimizers to perform global optimization:** As currently described, the algorithm finds a locally optimal policy, but not necessarily a globally optimal policy. For example, the stored states could be divided into two sets of nearest neighbors. One set could have a suboptimal policy, but have no interaction with the other set of states that had a globally optimal policy since no nearest neighbor relations joined the two sets. We expect the locally optimal policies to be fairly good because we 1) gradually increase the solved volume and 2) use local optimizers. Given local optimization of actions, gradually increasing the solved volume will result in a globally optimal policy if the boundary of this volume never touches a non adjacent section of itself. Figures 2 and 2 show the creases in the value function (discontinuities in the spatial derivative) and corresponding discontinuities in the policy that typically result when the constant cost contour touches a non adjacent section of itself as $V_{limit}$ is increased.

In theory, the approach we have described will produce a globally optimal policy if it has infinite resolution and all the stored states form a densely connected set in terms of nearest neighbor relations [8]. By enforcing consistency of the local value function models across all nearest neighbor pairs, we can create a globally consistent value function estimate. Consistency means that any state's local model correctly predicts values of nearby states. If the value function estimate is consistent everywhere, the Bellman equation is solved and we have a globally optimal policy. We can enforce consistency of nearest neighbor value functions by 1) using the policy of one state of a pair to reoptimize the trajectory of the other state of the pair and vice versa, and 2) adding more stored states in between nearest neighbors that continue to disagree [8]. This approach is similar to using the method of characteristics to solve partial differential equations and finding value functions for games.

In practice, we cannot achieve infinite resolution. To increase the likelihood of finding a globally optimal policy with a limited resolution of stored states, we need an analog to exploration and to global minimization with respect to actions found in the Bellman equation. We approximate this process by periodically reoptimizing each stored state using the policies of other stored states. As more and more states are stored, and many alternative stored states are considered in optimizing any given stored state, a wide range of actions are considered for each state. We run a reoptimization phase of the algorithm after every N (typically 100) states have been stored. There are several ways to design this reoptimization phase. Each state could use the policy of a nearest neighbor, or a randomly chosen neighbor with the distribution being distance dependent, or just choosing another state randomly with no consideration of distance (what we currently do). [8] describes how to follow a policy of another stored state if its trajectory is stored, or can be recomputed as needed. In this work we explored a different approach that does not require each stored state to save its trajectory or recompute it. To "follow" the policy of another state, we follow the locally linear policy for that state until the trajectory begins to go away from the state. At that point we switch to following the globally approximated policy. Since we apply this reoptimization process periodically with different randomly selected policies, over time we explore using a wide range of actions from each state.

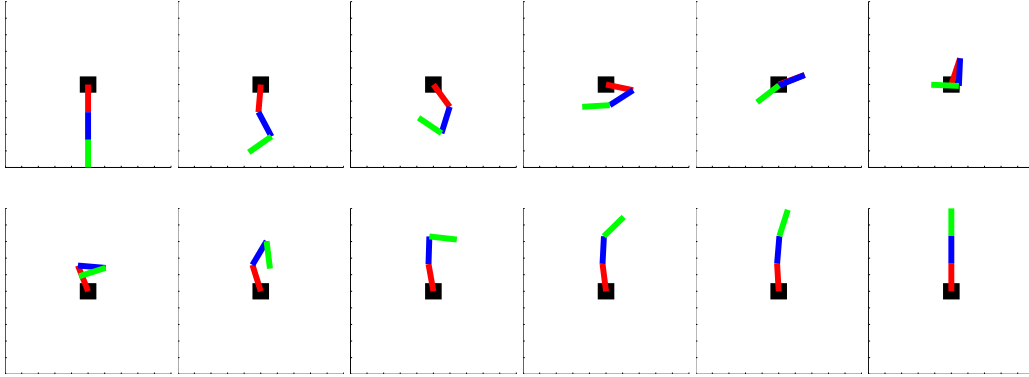

Figure 4: Configurations from the simulated three link pendulum optimal trajectory every tenth of a second and at the end of the trajectory.

## 4   Results

In addition to the one link swingup example presented in the introduction, we present results on two link swingup (4 dimensional state) and three link swingup (6 dimensional state). A companion paper using these techniques to explore how multiple balance strategies can be generated from one optimization criterion is [19]. Further results, including some for a four link (8 dimensional state) standing robot are presented.

**One link pendulum swingup:** For the one link swingup case, the random state approach found a globally optimal trajectory (the same trajectory found by our grid based approaches [15]) after adding only 63 random states. Figure 2:right shows the distribution of states and their trajectories superimposed on a contour map of the value function for one link swingup.

**Two link pendulum swingup:** For the two link swingup case, the random state approach finds what we believe is a globally optimal trajectory (the same trajectory found by our grid based approaches [15]) after storing an average of 12000 random states (Figure 3). In this case the state has four dimensions (a position and velocity for each joint) and a two dimensional action (a torque at each joint). The one step cost function was a weighted sum of the squared position errors and the squared torques: $L(\mathbf{x},\mathbf{u}) = 0.1(\theta_1^2 + \theta_2^2)T + (\tau_1^2 + \tau_2^2)T$. 0.1 weights the position errors relative to the torque penalty, T is the time step of the simulation (0.01s), and there were no costs associated with joint velocities. The approximately 12000 sampled states should be compared to the millions of states used in grid-based approaches. A 60x60x60x60 grid with almost 13 million states failed to find a trajectory as good as this one, while a 100x100x100x100 grid with 100 million states did find the same trajectory. In 13 runs with different random number generator seeds, the mean number of states stored at convergence is 11430. All but two of the runs converged after storing less than 13000 states, and all runs converged after storing 27000 states.

**Three link pendulum swingup:** For the three link swingup case, the random state approach found a good trajectory after storing less than 22000 random states (Figure 4). We have not yet solved this problem a sufficient number of times to be convinced this is a global optimum, and we do not have a solution based on a regular grid available for comparison. We were not able to solve this problem using regular grid-based approaches due to limited state resolution: 22x22x22x22x38x44 = 391,676,032 states filled our largest memory. As in the previous examples, the one step cost function was a weighted sum of the squared position errors and the squared torques: $L(\mathbf{x},\mathbf{u}) = 0.1(\theta_1^2 + \theta_2^2 + \theta_3^2)T + (\tau_1^2 + \tau_2^2 + \tau_3^2)T$.

## 5   Conclusion

We have combined random sampling of states and local trajectory optimization to create a promising approach to practical dynamic programming for robot control problems. We are able to solve problems we couldn't solve before due to memory limitations. Future work will optimize aspects and variants of this approach.

**Acknowledgments**

This material is based upon work supported in part by the DARPA Learning Locomotion Program and the National Science Foundation under grants CNS-0224419, DGE-0333420, ECS-0325383, and EEC-0540865.

# References

[1] J. Rust. Using randomization to break the curse of dimensionality. *Econometrica*, 65(3):487–516, 1997.

[2] M. Hauskrecht. Incremental methods for computing bounds in partially observable Markov decision processes. In *Proceedings of the 14th National Conference on Artificial Intelligence (AAAI-97)*, pages 734–739, Providence, Rhode Island, 1997. AAAI Press / MIT Press.

[3] N.L. Zhang and W. Zhang. Speeding up the convergence of value iteration in partially observable Markov decision processes. *JAIR*, 14:29–51, 2001.

[4] J. Pineau, G. Gordon, and S. Thrun. Point-based value iteration: An anytime algorithm for POMDPs. In *International Joint Conference on Artificial Intelligence (IJCAI)*, 2003.

[5] T. Smith and R. Simmons. Heuristic search value iteration for POMDPs. In *Uncertainty in Artificial Intelligence*, 2004.

[6] M.T.J. Spaan and Nikos V. A point-based POMDP algorithm for robot planning. In *Proceedings of the IEEE International Conference on Robotics and Automation*, pages 2399–2404, New Orleans, Louisiana, April 2004.

[7] S. Thrun. Monte Carlo POMDPs. In S.A. Solla, T.K. Leen, and K.-R. Müller, editors, *Advances in Neural Information Processing 12*, pages 1064–1070. MIT Press, 2000.

[8] C. G. Atkeson. Using local trajectory optimizers to speed up global optimization in dynamic programming. In Jack D. Cowan, Gerald Tesauro, and Joshua Alspector, editors, *Advances in Neural Information Processing Systems*, volume 6, pages 663–670. Morgan Kaufmann Publishers, Inc., 1994.

[9] F. L. Lewis and V. L. Syrmos. *Optimal Control, 2nd Edition*. Wiley-Interscience, 1995.

[10] C. Szepesvári. Efficient approximate planning in continuous space Markovian decision problems. *AI Communications*, 13(3):163–176, 2001.

[11] J. N. Tsitsiklis and Van B. Roy. Regression methods for pricing complex American-style options. *IEEE-NN*, 12:694–703, July 2001.

[12] V. D. Blondel and J. N. Tsitsiklis. A survey of computational complexity results in systems and control, 2000.

[13] R. Munos and A. W. Moore. Variable resolution discretization in optimal control. *Machine Learning Journal*, 49:291–323, 2002.

[14] S. M. LaValle. *Planning Algorithms*. Cambridge University Press, 2006.

[15] C. G. Atkeson. Randomly sampling actions in dynamic programming. In *2007 IEEE International Symposium on Approximate Dynamic Programming and Reinforcement Learning (ADPRL)*, 2007.

[16] C. G. Atkeson and J. Morimoto. Nonparametric representation of a policies and value functions: A trajectory based approach. In *Advances in Neural Information Processing Systems 15*. MIT Press, 2003.

[17] P. Dyer and S. R. McReynolds. *The Computation and Theory of Optimal Control*. Academic Press, New York, NY, 1970.

[18] D. H. Jacobson and D. Q. Mayne. *Differential Dynamic Programming*. Elsevier, New York, NY, 1970.

[19] C. G. Atkeson and B. Stephens. Multiple balance strategies from one optimization criterion. In *Humanoids*, 2007.

